# EEG2Video: Towards Decoding Dynamic Visual Perception from EEG Signals

**Xuan-Hao Liu**[1][*][†], **Yan-Kai Liu**[1][*], **Yansen Wang**[2][‡], **Kan Ren**[3][‡], **Hanwen Shi**[1],
**Zilong Wang**[2], **Dongsheng Li**[2], **Bao-Liang Lu**[1], **Wei-Long Zheng**[1][‡]
[1]Shanghai Jiao Tong University    [2]Microsoft Research Asia    [3]ShanghaiTech University
https://bcmi.sjtu.edu.cn/home/eeg2video

## Abstract

Our visual experience in daily life are dominated by dynamic change. Decoding such dynamic information from brain activity can enhance the understanding of the brain's visual processing system. However, previous studies predominately focus on reconstructing static visual stimuli. In this paper, we explore to decode dynamic visual perception from electroencephalography (EEG), a neuroimaging technique able to record brain activity with high temporal resolution (1000 Hz) for capturing rapid changes in brains. Our contributions are threefold: Firstly, we develop a large dataset recording signals from 20 subjects while they were watching 1400 dynamic video clips of 40 concepts. This dataset fills the gap in the lack of EEG-video pairs. Secondly, we annotate each video clip to investigate the potential for decoding some specific meta information (*e.g.*, color, dynamic, human or not) from EEG. Thirdly, we propose a novel baseline **EEG2Video** for video reconstruction from EEG signals that better aligns dynamic movements with high temporal resolution brain signals by Seq2Seq architecture. EEG2Video achieves a 2-way accuracy of 79.8% in semantic classification tasks and 0.256 in structural similarity index (SSIM). Overall, our works takes an important step towards decoding dynamic visual perception from EEG signals.

## 1 Introduction

Our visual experience are composed of continuously evolving scenes caused by the movement of objects and viewing perspective [1]. The intricate and complex visual system in our brains which enables us to explore the wonderful and ever-changing visual world has been appealing interests from philosophers and scientists for centuries [2–6]. To investigate the mechanism of our visual system, various neuroimaging techniques have been used to analyze brain activities, especially non-invasive methods like functional Magnetic Resonance Imaging (fMRI) [2, 7, 8], magnetoencephalography(MEG) [9–11], and electroencephalography(EEG) [12, 13].

Compared to fMRI and MEG which need to be recorded by large and expensive medical devices, EEG is relatively low-cost and portable and thus has been applied across many human visual studies [14–18]. For instance, a recent work achieved an accuracy of 15.6% in 200-way zero-shot tasks on an EEG-image dataset [19], demonstrating the rich visual information in EEG signals. However, these studies adopt either artificial strobe (SSVEP) [14, 15] or static image [16–18] as stimulation, which is far different from the dynamic visual world and not suitable for studying brain activities in naturalistic paradigm. As far as we know, there is currently no research on decoding video from EEG

---

[*]Xuan-Hao Liu and Yan-Kai Liu contribute equally.

[†]The work was conducted during Xuan-Hao Liu's internship at Microsoft Research Asia.

[‡]Corresponding authors: yansenwang@microsoft.com, renkan@shanghaitech.edu.cn, weilong@sjtu.edu.cn

signals. Consequently, people have limited knowledge about **1)** whether can we decode video from EEG signals? **2)** if yes, what kind of visual information can we decode?

In order to fill the gap, we develop a large EEG dataset, called **S**JTU **EEG D**ataset for **D**ynamic **Vis**ion (SEED-DV) dataset, collected from 20 subjects while they were watching a series of natural videos belonging to 40 different concepts. Also, we annotate some meta information for each video clip to comprehensively explore the boundary of which visual information can be decoded from EEG signals, offering a benchmark containing various visual decoding tasks across object recognition, color/motion perception, and human/face detection.

Besides the fundamental classification tasks, reconstructing visual perceptions from corresponding brain signals helps to advance the understanding of our visual neural system. With the development of the representation learning and artificial intelligence generated content (AIGC), numerous works have reconstructed vivid images from brain activities [20–30], which utilize the text-to-image generation models pretrained on large amounts of visual-language pairs by aligning the brain signals with corresponding text embeddings. Recently, some works reconstruct high-quality two-second videos from a single fMRI data frame [31, 32]. However, limited by the low temporal resolution of fMRI, these video generation frameworks lack the ability of capturing high dynamic changes.

To this end, we propose a novel baseline, EEG2Video, for video reconstruction from EEG signals based on Seq2Seq architectures which extract continuous low-level dynamic visual perception such as color and position from the brain signals of high temporal resolution. Afterwards, a dynamic-aware noise-adding (DANA) method is adopted for the diffusion process according to the dynamic information decoded from EEG. At last, we adopt the inflated diffusion model [33] fine-tuned on our dataset for video generation using the semantic information predicted from EEG. Our method densely extract visual information from high temporal resolution brain signals thus can better recover fast changes. Overall, our video reconstruction results take an important step towards decoding dynamic visual perception from EEG.

In conclusion, our contribution are as follows:

- For the first time, we develop a large EEG dataset named SEED-DV dataset collected from 20 subjects, offering 1400 EEG-video pairs from 40 concepts for studying dynamic visual information in EEG signals.
- We annotate the meta information of each video clip for comprehensively analyzing the visual information in EEG, presenting the EEG-VP benchmark.
- We evaluate various EEG models on the EEG-VP benchmark to determine the decoding ability of different visual information in raw EEG signals and human-extracted features.
- We propose a novel framework named EEG2Video for video reconstruction from EEG signals based on Seq2Seq architecture to densely utilize the highly dynamic information.
- The ablation study showcases the effectiveness of Seq2Seq and DANA modules in EEG2Video, which are designed based on the decoding results of different visual information on the EEG-VP benchmark.

## 2    Related Work

### 2.1    Decoding Static Visual Perception from Brain Activities

Researchers have been trying to decode low-level static visual perception (e.g., shape, color, and position) from brain activities for decades [34–37], revealing the abundant visual information hidden in brain signals. Early approaches tried to generate hazy silhouette or higher-quality images with Deconvolutional Neural Networks (DeCNNs)[38, 39], Generative Adversarial Networks (GANs)[40–44], and Variational Autoencoders (VAEs) [44]. Recent studies have obtained impressive results on decoding static visual stimuli from various brain activities like fMRI [20–27], MEG [28], and EEG [29, 30] with the Latent Diffusion Models (LDMs), also named Stable Diffusion (SD) [45–47], as image generation module, which is pretrained on a large-scale image dataset to generate vivid images based on text prompts. Specifically, text prompts are embedded into a text-image sharing latent space by CLIP [48], a multi-modal encoder to align visual and language representations. By encoding brain signals into the same sharing space, these methods can decode static visual perception with high diversity and fidelity using LDMs.

Compared to the high spatial resolutions of brain signals recorded by large devices like fMRI ($\approx$ 100,000 voxels) and MEG ($\approx$ 300 sensors), the limited spatial resolution of EEG ($\approx$ 60 sensors) brings difficulties to decode accurate visual perception from EEG in both semantic and pixel levels. Although previous studies claimed to achieve over 60% of semantic decoding accuracy on an EEG dataset with 40 classes [29, 30], the dataset[49] was blamed due to the block design [50]. The follow-up rigorous experiment [51] by randomly arranging all images showed at most 7.0% classification accuracy of the EEG siganls (chance level is 2.5%), exposing the fact that it is still challenging and insufficiently supported by appropriate datasets to decode visual perception from EEG.

## 2.2 Decoding Dynamic Visual Perception from Brain Activities

Observing the great success in decoding static visual perception, many endeavours have been devoted to decoding dynamic visual perception from fMRI [52–54, 31, 32]. These works were conducted on an fMRI-video dataset collected from 3 female subjects while watching a series of videos, including animals, humans, and natural scenery [52]. Due to the data sparsity, the DeCNN, VAE and GAN-based methods can only decode hazy perception of dynamic videos from fMRI [52–54]. To generate high-quality videos, MinD-Video[31] and NeuroCine[32] utilized an inflated SD model [33] for video generation, which incorporates network temporal inflation by adding temporal attention modules in the original SD to ensure the consistency between frames. The video SD is firstly fine-tuned using the text-video pairs in the training dataset (the text prompts were generated by an automatic image caption model called BLIP [55]), then the pre-trained fMRI encoder were co-trained with the video SD to enhance the fMRI guidance.

However, fMRI intrinsically lacks the ability of capturing dynamic visual perception due to the low temporal resolution, which is limited by the time scale of blood flow and results in a single fMRI frame every two seconds. MinD-Video[31] and NeuroCine[32] all decoded two-second videos from only one fMRI frame. Thus, they are unable to decode changes faster than 0.5 Hz. Instead, other neuroimaging techniques such as EEG with high temporal resolution up to thousands Hz can offer more appropriate alternatives. To the best of our knowledge, there is no such dataset studying decoding video stimuli from such signals, and we are the first to support this research direction with the dataset, benchmarks, and the decoding framework.

# 3 EEG Dynamic Vision Dataset and Benchmarks

In this section, we first describe how we construct the **S**JTU **EEG D**ataset for **D**ynamic **V**ision (SEED-DV). Then we introduce two benchmarks built on the SEED-DV dataset: EEG visual perception classification benchmark and video reconstruction benchmark. The purpose of building this new dataset is to answer the following research questions:

**RQ1** : Whether can we decode dynamic visual information from EEG signals?

**RQ2** : If yes, which visual information can be decoded?

**RQ3** : To what extent can we reconstruct video from EEG signals?

Hence, we carefully selected video clips suitable for studying dynamic vision and annotated their meta information.

## 3.1 Participants

Twenty healthy students from Shanghai Jiao Tong University (SJTU) participated (mean age: 21.75 STD: 2.05; 10 females, 10 males), all having normal or corrected-to-normal vision. All subjects were informed of the experimental process and signed informed consent forms before the experiment, then received monetary reimbursement after finishing. This study was approved by the ethical committee of SJTU Institutional Review Board for Human Research Protections.

## 3.2 Visual Stimuli Selection

We elaborately selected 40 concepts of videos in our experiment to study. The number of concepts follows previous research on EEG-image pairs [51, 49]. It is worth noting that EEG-Things [13]

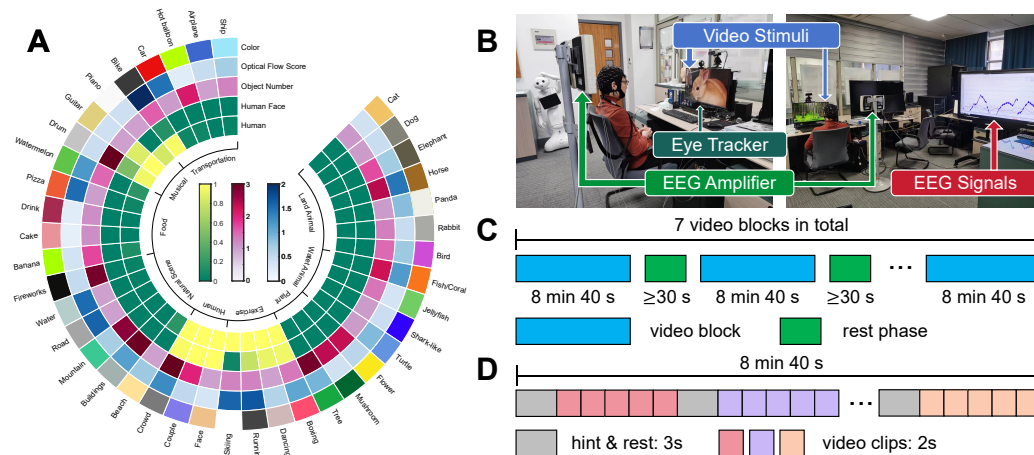

Figure 1: The meta information of video clips of 40 concepts, and experiment protocol. (A) Visualizations of the meta information for all video clips of 40 concepts, we plot the average of each meta information for each concept. (B) The data collecting environment. (C) Demonstration of a whole data collecting session. A session will contain 7 video blocks to be watched, and there are rest phases of at least 30 seconds each between blocks. (D) Demonstration of a video block, there is a 3-second hint before 5 different video clips of the same concept.

dataset (with 1854 object concepts) employed a rapid serial visual presentation (RSVP) paradigm [56, 57], which cannot be used for EEG-video experiment as video is a continuous stimulation. We further cluster the concepts into 9 coarser classes, as is demonstrated in Figure 1(A): {*land animal, water animal, plant, exercise, human, natural scene, food, musical instrument, transportation*}.

All video clips are collected from two online video websites, Bilibili[1] and YouTube[2]. We selected 35 different two-second video clips for each concept, which are divided randomly into 7 groups, each group has 5 video clips. Afterwards, 40 groups of different concepts are arranged sequentially to form a single block, each block containing $40 \times 5 = 200$ video clips. Consequently, there are 7 blocks in our experiment, where the order of 40 categories of videos within each block is random and different from each other to mitigate the temporal bias of EEG signals.

## 3.3 Experiment Protocol

To ensure the quality of the acquired EEG data, the experiments were conducted in a controlled laboratory environment to minimize noise and other environmental disturbances. 62-channel EEG signals were collected by an active AgCl electrode cap with an international 10-10 system. The EEG signals were acquired using the ESI NeuroScan System at a sampling rate of 1000 Hz. Besides EEG data, EOG and ECG signals were recorded simultaneously during the experiment. We also adopt a Tobii Pro Fusion eye tracker to collect eye movements at a sampling rate of 250Hz.

During the experiment, all subjects were instructed to watch a series of color video clips presented with the resolution of $1980 \times 1080$ (16:9) in full-screen mode on a 25-inch display. In each block, the 5 video clips with the same category were displayed continuously, and before playing these 5 same-class videos, there is a hint on the screen to inform the subjects what class they will see next, which will last for three seconds. Consequently, there are $40 \times (3 + 5 \times 2) = 520$ seconds, i.e., 8 min 40 s for each block.

There are 7 blocks of videos to watch in our experiment for a subject. After finishing watching a block (40 classes), the subject was required to rest for at least 30 seconds to mitigate fatigue and assesses his/her own attention level (ranging from 1 to 5. 1 means sleepy and 5 means very concentrating)

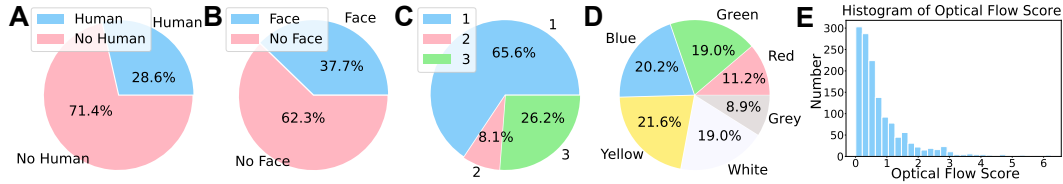

Figure 2: Statistics of each meta information: (A) the fraction of human appearance. (B) the fraction of face appearance (only count the videos with humans). (C) the distribution of different object numbers. (D) the distribution of different object colors. (E) the histogram of OFS.

before starting the next block. As a result, the average attention level across all the subjects and blocks is $4.01 \pm 0.83$, ensuring high-quality EEG signals with acceptable concentrations.

### 3.4 Meta Information of Video Clips

Besides the 40 concepts and 9 coarser classes, we also labeled some other meta information for each video clip.

**Color**: The main color of the main object. There are 7 color categories: {*Red*, *Yellow*, *Blue*, *Green*, *White*, *Grey*, *Colorful*}. *Colorful* indicates the color is too complex for identifying an accurate color from a single video clip.

**Optical Flow Score**: The optical flow score (OFS) of each 24 FPS two-second video clip obtained by averaging the length of the optical flow vectors, ranging from 0.008 (almost static) to 6.252 (rapidly changing). Further, based on the OFS, we divide all the video clips into 2 categories: {*Fast*, *Slow*}. We choose the median OFS of 1.799 as the threshold to make sure the label is balanced.

**Object Number**: The number of the main objects. There are 3 categories: {*One*, *Two*, *Many*}. *Many* indicates the number of the main objects is equal to or more than three.

**Human**: If there are any humans appearing in the video, the label is *1*, otherwise is *0*.

**Human Face**: If there are any human faces appearing in the video, the label is *1*, otherwise is *0*.

We depict the average meta information of each concept in Figure 1(A):

### 3.5 EEG Visual Perception Classification Benchmark

To better understand the visual information in EEG signals, some low-level and high-level visual perception tasks are investigated in our dataset, presenting an EEG Viusal Perception (EEG-VP) benchmark. There are 7 EEG classification tasks based on the video label and meta information detailed in Section 3.4, whose statistics are presented in Figure 2:

- The 40-class classification of the fine-grained concept of the video clip.
- The 9-class classification of the course concept of the video clip.
- The 6-class classification of which color of the main object in the video clip. The data while watching *Colorful* videos are discarded in this task.
- The binary classification of whether the video is *fast* or *slow* based on the OFS.
- The 3-class classification of the number of the main objects in the video clip.
- The binary classification of whether human appears in the video or not.
- The binary classification of whether any human face appears in the video or not. The data while watching videos without human appearance are discarded in this task.

### 3.6 Video Reconstruction Benchmark

This task is to reconstruct two-second video clips from the corresponding EEG segments. Following the previous video reconstruction from fMRI studies [31, 32, 52–54], we utilize the same metrics to

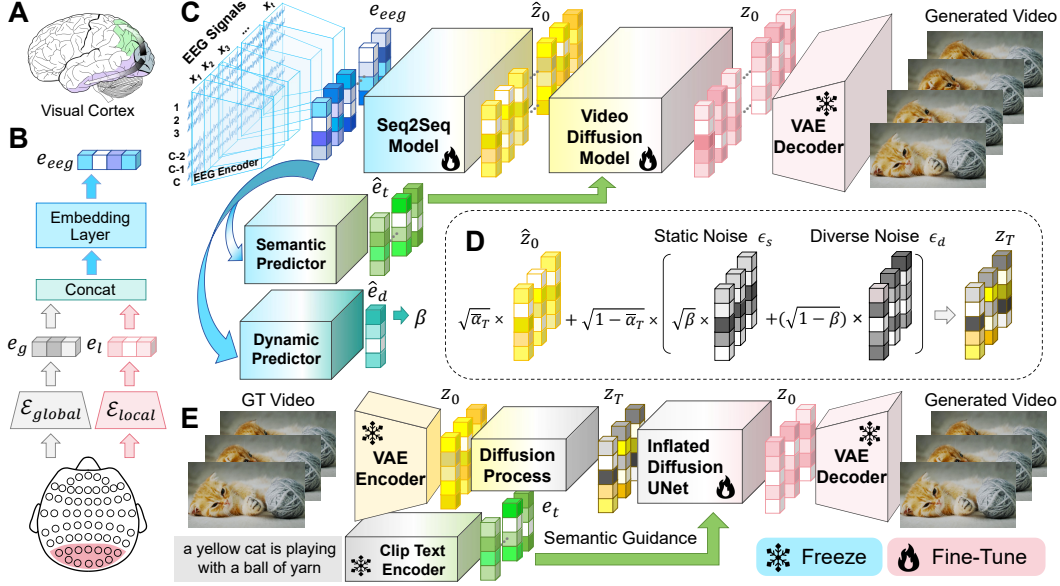

Figure 3: (A-B) GLMNet Encoders. (C-E) Overview of our EEG2Video framework. (A) The visual cortex, basically in the occipital lobe. (B) GLMNet architecture which combines the global and local embedding. (C)The framework of EEG2Video which predicts the latent variabels $\hat{z}_0$ and semantic guidance $\hat{e}_t$ with Seq2Seq model and a predictor, respectively. A video diffusion model is then be adopted for generating videos using $\hat{z}_0$ and $\hat{e}_t$. (D) Dynamic-aware noise-adding process based on the decoded dynamic information $\beta$. (E) Using large amounts of video-text pairs to fine-tune the inflated diffusion UNet for video generation. The texts are obtained by BLIP.

evaluate the quality of generated videos, roughly classified as frame-based metrics and video-based metrics. Definition of these metrics can be found in Appendix A.

# 4 Methodology

## 4.1 EEG Encoder: Mixture of Global and Local Features.

EEG classifiers take into an EEG segment $x \in \mathcal{X} \subset \mathbb{R}^{C \times T}$, where $C$ denotes EEG channels and $T$ denotes time samples, and decode the target information $y \in \mathcal{Y}$, which can be defined as $f : \mathcal{X} \to \mathcal{Y}$.

Previous studies on EEG classification always treat all channels equally [49, 58, 59]: $f(x) = \mathcal{E}_{global}(x)$, where $\mathcal{E}_{global}$ is a global encoder for decoding EEG. However, research in neuroscience indicates that human visual cortex are basically in the occipital lobe, as shown in Figure 3(A), treating all channels equally is unable to highlight the visual cortex features. To this end, we propose a simple yet effective network to combine the global features decoded from whole channels and local features decoded from the visual associated channels called Global Local Mixture Network (GLMNet). Depicted in Figure 3(B), GLMNet utilizes a local encoder $\mathcal{E}_{local}$ to extract the vision-associated features, which can be denoted as $f(x) = emb(\texttt{Concat}(\mathcal{E}_{global}(x), \mathcal{E}_{local}(x)))$.

## 4.2 EEG2Video: High Temporal Resolution Brain Decoding Framework

Compared to the fMRI-video reconstruction framework [31, 32] which are constrained by the limited temporal resolution that results in decoding two-second dynamic videos from one fMRI data frame, EEG signals have higher temporal resolution for better capturing dynamic visual perception to facilitate video reconstruction. As demonstrated in Figure 3(C), we introduce **EEG2Video**, a novel framework which utilize a Seq2Seq model to densely reconstruct low-level visual perception from continuous EEG embeddings extracted by an overlapping sliding window, and decode the semantic

and dynamic information with two other modules for guiding an inflated diffusion model to recreate videos.

**EEG Embeddings Extraction** For a two-second EEG segment $x \in \mathbb{R}^{C \times T}$, we apply an overlapping sliding window for slicing it into shorter segments $\{x_1, x_2, \ldots, x_t\}$, where $t$ is the total number. EEG embeddings $e_{eeg} = \{e_e^1, e_e^2, \ldots, e_e^t\}$ are then extracted from these shorter segments with an EEG encoder $\mathcal{E}$ that $\mathcal{E}(x_i) = e_e^i, i = \{1, 2, \ldots, t\}$.

**Seq2Seq Model** In contrast to video reconstruction from fMRI (generating several frames from a single data frame), it is essential to align the high temporal resolution brain signals with videos in video reconstruction from EEG (generating several frames from several EEG segments) for capturing rapid changes. The Seq2Seq models are naturally introduced for extracting the continuous visual information from high temporal resolution brain signals. We employ the Transformer architecture as the Seq2Seq model in our framework, which can be formulated as a stack of several blocks, each block containing a multi-head attention (MHA) layer and a feed-forward network (FFN) layer[60]. Denoting the input of the $i$-th Transformer block as $x_{\text{in}}^i$, the calculation of the output $x_{\text{out}}^i$ is given by:

$$x_{\text{mid}}^i = f_{\text{MHA}}(\text{LA}(x_{\text{in}}^i)) + x_{\text{in}}^i, \ \ x_{\text{out}}^i = f_{\text{FFN}}(\text{LA}(x_{\text{mid}}^i)) + x_{\text{mid}}^i, \tag{1}$$

where $f_{\text{MHA}}$ is the MHA layer, $f_{\text{FFN}}$ is the FFN layer, and LA is layer normalization. In our framework, the input of Transformer is the addition of EEG embeddings and position embeddings (PE): $x_{\text{in}}^0 = e_{eeg} + PE$, and the output is the latent variables $\hat{z}_0$ of the corresponding video frames. As depicted in Figure 3(E), the Ground Truth (GT) video frames are fed into the frozen VAE encoder to obatin the GT latent variables $z_0$. We apply mean squared error (MSE) loss $MSE(\hat{z}_0, z_0)$ for training the Seq2Seq model that densely predicts the continuous visual information of frames.

**Semantic Predictor** In order to utilize the pre-trained diffusion models for generating high-quality videos, we first generate the corresponding text description of each video by feeding the medium frame to a caption model called BLIP [55], then align the EEG signals with the text embeddings $e_t \in \mathbb{R}^{77 \times 768}$, which are acquired by the frozen CLIP text encoder as shown in Figure 3(E). An MLP is adopted as the semantic predictor to project EEG data into the same dimension to obtain $\hat{e}_t$. Finally, the MSE loss $MSE(\hat{e}_t, e_t)$ is employed for aligning EEG and text embeddings.

**Dynamic-Aware Noise-Adding Process** The frames in a video with high OFS are more diverse from each other than those in videos with low OFS. Based on the dynamic decoding results, we can roughly classify whether the video is high dynamic or not from EEG signals. Hence, we introduce the static noise $\epsilon_s$ and diverse noise $\epsilon_d$ into the diffusion process and balance them by the decoded dynamic information $\hat{e}_d$. The diverse noise $\epsilon_d = \{\epsilon_d^1, \epsilon_d^2, \ldots, \epsilon_d^n\}$ consists of $n$ different noises, each $\epsilon_d^i \sim \mathcal{N}(0, 1)$. The static noise $\epsilon_s$ has the same noise $\epsilon \sim \mathcal{N}(0, 1)$ replicated $n$ times. The ratio of the static noise is smaller when the $\hat{e}_d$ indicates the video is more dynamic. The diffusion process can be formulated as follows to acquire the noise $z_{\mathcal{T}}$ at time steps $\mathcal{T}$:

$$z_{\mathcal{T}} = \sqrt{\overline{\alpha}_{\mathcal{T}}} \times \hat{z}_0 + \sqrt{1 - \overline{\alpha}_{\mathcal{T}}} \times (\sqrt{\beta} \times \epsilon_s + \sqrt{1 - \beta} \times \epsilon_d), \tag{2}$$

where $\overline{\alpha}_{\mathcal{T}}$ is the coefficient for adding noise directly to the noise at time steps $\mathcal{T}$, $\beta$ is depended by the decoded dynamic information. Here, We set $\beta = 0.2$ when the video is high dynamic and $\beta = 0.3$ when the video is low dynamic.

**Video Diffusion Model** For reconstructing vivid videos from EEG signals, we utilize the Tune-A-Video technique which fine-tunes an inflated text-to-image diffusion model [33]. The network inflation trick is adding a sparse temporal attention layer in the image generation model to ensure the consistency between frames, in which each frame is calculated with the the first frame and the frame before it. Using the same notations in [33], the attention is formulated as:

$$Q = W^Q \cdot z_{v_i}, \ \ K = W^K \cdot [z_{v_{i-1}}, z_{v_1}], \ \ V = W^V \cdot [z_{v_{i-1}}, z_{v_1}], \tag{3}$$

where $[\cdot]$ denotes concatenation operation, and $z_{v_i}$ is the $i$-th frame. To fine-tune the video generation model in our framework, the corresponding text description of each video by feeding the medium frame to a caption model called BLIP [55]. Afterwards, all video-text pairs in the training set are used for fine-tuning the Stable Diffusion Model V1-4 [45].

# 5 Experiment on SEED-DV Dataset

In this section, we evaluate the performance of our methods and other baselines on the proposed two benchmarks: the EEG-VP benchmark and the video reconstruction benchmark. The details of data pre-processing, model implementation and training can be found in Appendix B.

## 5.1 EEG-VP Benchmark

### 5.1.1 Quantitative Results

We present the overall accuracy of different EEG classifiers in Table 1. Besides raw EEG Signals, we also run experiments on the PSD features and the DE features [61] of 5 frequency bands.

From the result, we can see that (1) our GLMNet outperformed the baselines consistently across all the classification tasks, which indicates the importance of extracting vision-associated features for visual perception tasks. (2) Different types of EEG features yield similar results. (3) Different tasks have different difficulties. Some meta information are distinguishable via EEG signals, e.g., *color*, *dynamics*, while the statistical significance indicates that the number of main object and whether *human/face* appear are difficult or even impossible to be classified. The difference may be attributed to different processing mechanism by our brains which can inspire research in neuroscience.

To conclude, we can now answer **RQ1** and **RQ2**: Yes, some of visual information can be decoded from EEG signals, including *category*, *color*, *dynamics*. However, the overall visual perception benchmark is rather challenging and calls for more advanced algorithms. Refer to Appendix C for the confusion matrix and analysis across subjects.

Table 1: Average classification accuracy (%) and std across all subjects with different EEG classifiers on different tasks. Chance level is the percentage of the largest class. The star symbol (*) represents the result is above chance level with statistical significance (two-sample t-test: $p < 0.05$).

| Methods | 40-c top-1 | 40-c top-5 | 9-c top-1 | 9-c top-3 | Color | Fast/Slow | Numbers | Human Face | Human |
|---|---|---|---|---|---|---|---|---|---|
| Chance level | 2.50 | 12.50 | 11.11 | 33.33 | 20.57 | 50.00 | 65.64 | 62.25 | 71.43 |
| | | | | Raw EEG Signals | | | | | |
| ShallowNet[62] | 5.59/2.27* | 16.93/4.66* | 21.40/1.96* | 49.62/2.34* | 27.00/2.09* | 56.62/1.77* | 66.15/0.89 | 64.87/1.54 | 73.21/1.52 |
| DeepNet[62] | 4.56/1.52* | 14.30/3.25* | 20.27/1.25* | 48.06/1.59* | 26.37/1.95* | 55.42/0.59* | 65.71/0.24 | 61.58/3.93 | 72.86/0.40 |
| EEGNet[58] | 4.64/0.86* | 14.25/1.87* | 19.63/0.81* | 47.04/1.45* | 25.46/1.31* | 51.99/2.00 | 64.67/0.60 | 61.37/1.31 | 72.38/0.98 |
| Conformer[59] | 4.93/1.57* | 15.36/4.44* | 20.92/0.98* | 49.25/1.49* | **27.53/1.37*** | 55.02/0.83* | 65.73/0.26 | 64.96/1.14 | 73.00/0.85 |
| TSConv[19] | 4.92/0.99* | 15.05/2.31* | 20.00/1.01* | 47.76/1.51* | 26.89/1.83* | 55.32/0.99* | 65.39/0.41 | 64.39/1.47 | 72.68/0.67 |
| GLMNet (Ours) | **6.20/3.02*** | **17.75/4.24*** | **21.93/1.87*** | **50.01/2.52*** | 27.33/1.45* | **57.35/1.98*** | **66.21/0.91** | **65.10/1.45** | **73.34/1.31** |
| | | | | PSD Features | | | | | |
| SVM[63] | 5.19/2.81* | - | 19.02/3.27* | - | 21.31/2.97 | 53.56/1.11* | 64.15/1.22 | 58.94/2.21 | 70.91/1.84 |
| MLP | 6.20/3.02* | 18.91/5.94* | 21.59/3.00* | 49.86/3.78* | 22.02/3.27 | 55.15/1.20* | 64.48/0.92 | 63.94/1.13 | 71.74/1.76 |
| GLMNet (Ours) | **6.23/2.91*** | **18.98/5.62*** | **21.69/3.20*** | **50.03/4.10*** | **26.40/2.99*** | 55.42/1.32* | **64.68/0.92** | **64.22/1.43** | **72.27/1.57** |
| | | | | DE Features | | | | | |
| SVM[63] | 4.82/2.80* | - | 19.05/3.39* | - | 21.07/2.88 | 53.34/1.25* | 63.62/1.73 | 57.82/3.50 | 70.25/1.94 |
| MLP | 6.12/3.08* | 19.02/5.71* | 21.17/3.24* | 49.40/4.94* | 25.91/3.27 | 54.76/1.25* | 64.10/0.70 | 63.41/1.57 | 71.74/1.76 |
| GLMNet (Ours) | **6.16/3.18*** | **19.12/6.07*** | **21.34/3.34*** | **49.55/4.57*** | **26.15/3.24*** | 55.06/1.20* | 64.25/0.74 | 63.63/1.80 | **72.27/1.58** |

### 5.1.2 Analysis of Brain Areas

To find electrodes or brain areas most associated with dynamic visual perception, we conduct a one-channel classification task to test the classification quality of each electrode. Due to the reason that only one channel is used, we simplify the task to binary classification: Human/Animal and Fast/Slow tasks, which is related to object recognition and dynamic perception, respectively. It can be observed from Figure 4(A) that the electrodes in the occipital area have higher accuracy on Human/Animal tasks, demonstrating the object recognition are related to the occipital area where the visual cortex is located, presenting a similar result as previous works [64, 19, 30]. However, not all dynamic visual perception are in the occipital region. Figure 4(B) reveals that the brain area associ-

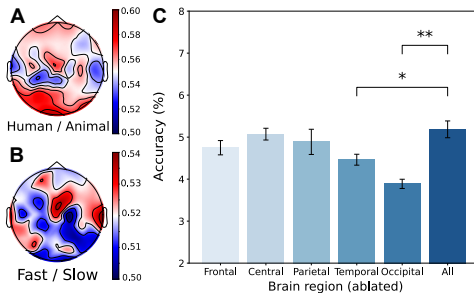

Figure 4: Spatial Analysis. (A-B). Topographies of each electrode's accuracy for Human/Animal and Fast/Slow tasks. (C). Ablate electrodes of different brain regions.

ated to movements are around the temporal region where the sensory and motor cortex lies, consistent with the previous neuroscience study [65].

To verify the findings, we conduct the ablation study by removing electrodes from different brain regions and show the 40-c top-1 accuracy in Figure 4(C). Removing occipital region significantly damages the performance ($p < 0.01$). The performance also declines without the temporal region.

## 5.2 Video Reconstruction

### 5.2.1 Quantitative Results

In this section, we evaluate our framework on the SEED-DV dataset and three subsets which contains less categories to answer **Q3**. The quantitative results under four cases are reported in Table 2. With the number of classes increases, the reconstruction performances decrease. Our framework achieves 34.0% of 40-way semantic-level accuracy when dealing with the subset contains 10 classes and 15.9% when facing the whole 40 classes. Meanwhile, the structural similarity index measure (SSIM) [66], which reflects the pixel-level similarity between reconstructed samples and ground truth samples, is up to 0.300 when facing the smallest subset, and drops to 0.256 when facing whole dataset. While the scores cannot be directly compared, our reconstructed videos have very similar and even higher SSIM to that reconstructed by fMRI reported in [32], intuitively demonstrating the capability of EEG signals to reconstruct dynamic visual perceptions. And it is also worth noting that with this more sophisticated model, the generated videos have much higher 40-way classification accuracy than that reported in EEG-VP benchmark, highlighting the potential to advance this research direction.

Based on the findings from the results of the EEG-VP benchmark (the statistic significance proves that *Category*, *Color*, and *Dynamics* can be decoded from EEG signals), we design the DANA module for injecting the *Fast/Slow* into diffusion process, the semantic predictor to inject *Category* information, and the general Seq2Seq for decoding low-level visual information like *Color*. We further conduct the ablation study by removing the Seq2Seq module and the DANA process respectively, and we can see huge performance drop without either module. This indicates that capturing the dynamics of both EEG and video is crucial for successful video reconstruction.

Table 2: Quantitative results of each methods on different size of subsets. Standard deviation is calculated across random seeds.

| # Classes | Metrics | Video-based | | Frame-based | | |
| --- | --- | --- | --- | --- | --- | --- |
| | | Semantic-level | | Semantic-level | | Pixel-level |
| | Models | 2-way | 40-way | 2-way | 40-way | SSIM |
| 10 | Full Model | $0.852\pm0.02$ | $0.340\pm0.01$ | $0.798\pm0.03$ | $0.232\pm0.02$ | $0.300\pm0.03$ |
| | w/o Seq2Seq | $0.772\pm0.02$ | $0.117\pm0.01$ | $0.696\pm0.02$ | $0.155\pm0.03$ | $0.187\pm0.03$ |
| | w/o DANA | $0.803\pm0.02$ | $0.183\pm0.01$ | $0.679\pm0.02$ | $0.092\pm0.01$ | $0.292\pm0.03$ |
| 20 | Full Model | $0.813\pm0.02$ | $0.273\pm0.03$ | $0.785\pm0.04$ | $0.184\pm0.02$ | $0.242\pm0.03$ |
| | w/o Seq2Seq | $0.800\pm0.02$ | $0.119\pm0.01$ | $0.685\pm0.02$ | $0.099\pm0.03$ | $0.187\pm0.04$ |
| | w/o DANA | $0.811\pm0.02$ | $0.240\pm0.03$ | $0.784\pm0.01$ | $0.164\pm0.02$ | $0.231\pm0.03$ |
| 30 | Full Model | $0.794\pm0.02$ | $0.209\pm0.05$ | $0.785\pm0.04$ | $0.180\pm0.02$ | $0.228\pm0.04$ |
| | w/o Seq2Seq | $0.770\pm0.02$ | $0.122\pm0.04$ | $0.741\pm0.02$ | $0.173\pm0.04$ | $0.207\pm0.03$ |
| | w/o DANA | $0.789\pm0.02$ | $0.160\pm0.02$ | $0.769\pm0.02$ | $0.139\pm0.01$ | $0.198\pm0.02$ |
| 40 | Full Model | $0.798\pm0.03$ | $0.159\pm0.01$ | $0.774\pm0.02$ | $0.138\pm0.01$ | $0.256\pm0.03$ |
| | w/o Seq2Seq | $0.786\pm0.03$ | $0.113\pm0.01$ | $0.734\pm0.02$ | $0.112\pm0.01$ | $0.189\pm0.03$ |
| | w/o DANA | $0.770\pm0.02$ | $0.128\pm0.01$ | $0.732\pm0.03$ | $0.109\pm0.03$ | $0.217\pm0.02$ |

### 5.2.2 Reconstructed Examples

We present some visual examples in Figure 5. It can be observed that various videos are reconstructed, and as long as the semantic and low-level visual information is correctly predicted by the model, the downstream diffusion process can generate vivid and high-quality videos. More examples including failure cases with full 6 frames can be found in Appendix D.

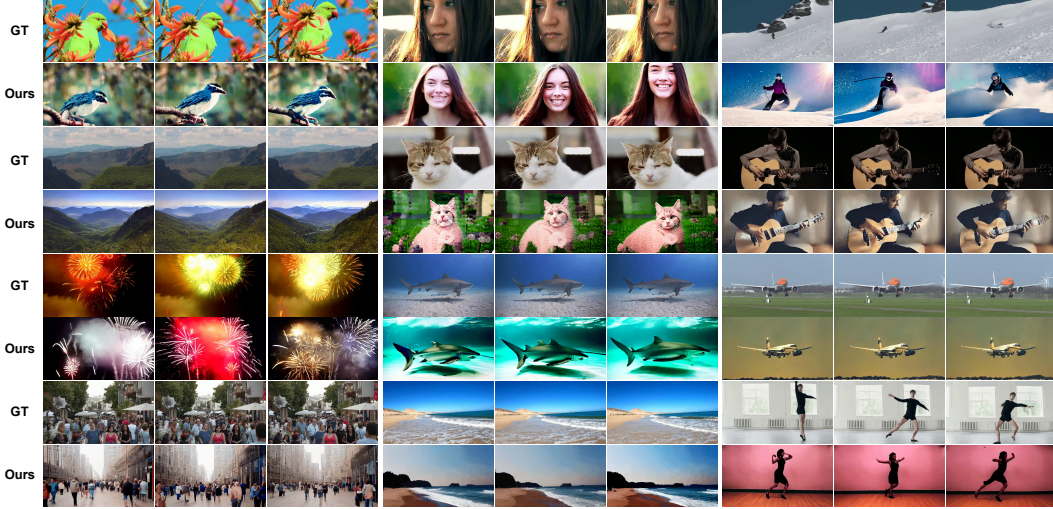

Figure 5: Reconstruction Presentations. Various video clips with low dynamics (e.g., *Mountain*, *Beach*, *Face*) and high dynamics (e.g., *Skiing*, *Fireworks*, *Dancing*) across animals, scenes, persons, and activities can be correctly recovered.

## 6 Conclusion

In this paper, we developed the large dataset, SEED-DV, to reconstruct videos from EEG signals, upon which we built the EEG Visual Perception Classification benchmark and the Video Reconstruction benchmark to support evaluating the advances of EEG-based video reconstruction. Moreover, we proposed a novel baseline EEG2Video for video reconstruction from EEG signals that can align visual dynamics with EEG based on the Seq2Seq architecture, and we presented vivid generated examples by training our framework on SEED-DV.

As the first attempt, we open a new possibility for BCI researchers to decode dynamic visual perception from EEG signals. Although the overall performance on SEED-DV are still in a preliminary stage, we hold the strong belief that the game-changing results for the BCI area can soon be discovered.

## 7 Broader Impacts

Reconstructing dynamic visual perception from brain activities helps to advance the understanding of our visual system in brains. EEG is a physiological signal widely used in clinical practice and brain-computer interfaces. Compared to non-portable and expensive neuroimaging techniques like fMRI and MEG, our work provides a convenient and cheap solution for decoding visual information from brain activities. This technique can be used for visualize our mind, offering a novel approach for listening the inner world of people patients with mental illnesses like autistic and depression.

However, every coin has two sides. Personal privacy may leak through our brain activities and be abused by malicious attackers for reading one's mind from EEG signals without acknowledgment. More strict regulations are supposed to be made for protecting the privacy of people's biological data by government and medical institutions.

## 8 Limitations

Our SEED-DV dataset currently records the EEG signals of each subject with one session, leading to the requirement for collecting more EEG-Video pairs with more sessions, which is significant for studying the stable neural patterns over time. Our framework is evaluated under the subject-dependent settings, the cross-subject ability remains unexplored due to individual variations. Future work can be focused on the transferability and stability of the video reconstruction framework by exploiting generalizable EEG encoders and Seq2Seq model.

## Acknowledgments

This work was supported in part by grants from National Natural Science Foundation of China (Grant No. 62376158), STI 2030-Major Projects+2022ZD0208500, Shanghai Municipal Science and Technology Major Project (Grant No. 2021SHZD ZX), Shanghai Pujiang Program (Grant No. 22PJ1408600), Medical-Engineering Interdisciplinary Research Foundation of Shanghai Jiao Tong University "Jiao Tong Star" Program (YG2023ZD25, YG2024ZD25 and YG2024QNA03), Shanghai Pilot Program for Basic Research - Shanghai Jiao Tong University (No. 21TQ1400203) and GuangCi Professorship Program of RuiJin Hospital Shanghai Jiao Tong University School of Medicine.

## Footnotes

[1] https://www.bilibili.com

[2] https://www.youtube.com

## References

[1] W. R. Hendee and P. N. Wells, *The perception of visual information*. Springer Science & Business Media, 1997.

[2] K. Grill-Spector and R. Malach, "The human visual cortex," *Annu. Rev. Neurosci.*, vol. 27, pp. 649–677, 2004.

[3] F. Tong, "Primary visual cortex and visual awareness," *Nature Reviews Neuroscience*, vol. 4, no. 3, pp. 219–229, 2003.

[4] D. H. Hubel and T. N. Wiesel, "Receptive fields of single neurones in the cat's striate cortex," *The Journal of Physiology*, vol. 148, no. 3, p. 574, 1959.

[5] H. Jang and F. Tong, "Improved modeling of human vision by incorporating robustness to blur in convolutional neural networks," *Nature Communications*, vol. 15, no. 1, p. 1989, 2024.

[6] S. Lees, N. Dayan, H. Cecotti, P. McCullagh, L. Maguire, F. Lotte, and D. Coyle, "A review of rapid serial visual presentation-based brain–computer interfaces," *Journal of neural engineering*, vol. 15, no. 2, p. 021001, 2018.

[7] S. A. Engel, D. E. Rumelhart, B. A. Wandell, A. T. Lee, G. H. Glover, E.-J. Chichilnisky, M. N. Shadlen *et al.*, "fmri of human visual cortex," *Nature*, vol. 369, no. 6481, pp. 525–525, 1994.

[8] K. Han, H. Wen, J. Shi, K.-H. Lu, Y. Zhang, D. Fu, and Z. Liu, "Variational autoencoder: An unsupervised model for encoding and decoding fmri activity in visual cortex," *NeuroImage*, vol. 198, pp. 125–136, 2019.

[9] F. Moradi, L. Liu, K. Cheng, R. A. Waggoner, K. Tanaka, and A. A. Ioannides, "Consistent and precise localization of brain activity in human primary visual cortex by meg and fmri," *Neuroimage*, vol. 18, no. 3, pp. 595–609, 2003.

[10] P. Adjamian, A. Hadjipapas, G. R. Barnes, A. Hillebrand, and I. E. Holliday, "Induced gamma activity in primary visual cortex is related to luminance and not color contrast: An meg study," *Journal of Vision*, vol. 8, no. 7, pp. 4–4, 2008.

[11] P. Ramkumar, B. C. Hansen, S. Pannasch, and L. C. Loschky, "Visual information representation and rapid-scene categorization are simultaneous across cortex: An meg study," *Neuroimage*, vol. 134, pp. 295–304, 2016.

[12] N. A. Busch, J. Dubois, and R. VanRullen, "The phase of ongoing eeg oscillations predicts visual perception," *Journal of neuroscience*, vol. 29, no. 24, pp. 7869–7876, 2009.

[13] A. T. Gifford, K. Dwivedi, G. Roig, and R. M. Cichy, "A large and rich EEG dataset for modeling human visual object recognition," *NeuroImage*, vol. 264, p. 119754, 2022.

[14] D. Regan, "Some characteristics of average steady-state and transient responses evoked by modulated light," *Electroencephalography and clinical neurophysiology*, vol. 20, no. 3, pp. 238–248, 1966.

[15] N. K. N. Aznan, S. Bonner, J. Connolly, N. Al Moubayed, and T. Breckon, "On the classification of ssvep-based dry-eeg signals via convolutional neural networks," in *2018 IEEE international conference on systems, man, and cybernetics (SMC)*. IEEE, 2018, pp. 3726–3731.

[16] S. Bagchi and D. R. Bathula, "Eeg-convtransformer for single-trial eeg-based visual stimulus classification," *Pattern Recognition*, vol. 129, p. 108757, 2022.

[17] D. Veniero, J. Gross, S. Morand, F. Duecker, A. T. Sack, and G. Thut, "Top-down control of visual cortex by the frontal eye fields through oscillatory realignment," *Nature communications*, vol. 12, no. 1, p. 1757, 2021.

[18] T. Grootswagers, I. Zhou, A. K. Robinson, M. N. Hebart, and T. A. Carlson, "Human EEG recordings for 1,854 concepts presented in rapid serial visual presentation streams," *Scientific Data*, vol. 9, no. 1, p. 3, 2022.

[19] Y. Song, B. Liu, X. Li, N. Shi, Y. Wang, and X. Gao, "Decoding natural images from EEG for object recognition," in *The Twelfth International Conference on Learning Representations*, 2024. [Online]. Available: https://openreview.net/forum?id=dhLIno8FmH

[20] Y. Takagi and S. Nishimoto, "High-resolution image reconstruction with latent diffusion models from human brain activity," in *Proceedings of the IEEE/CVF Conference on Computer Vision and Pattern Recognition (CVPR)*, 2023, pp. 14 453–14 463.

[21] Z. Chen, J. Qing, T. Xiang, W. L. Yue, and J. H. Zhou, "Seeing beyond the brain: Conditional diffusion model with sparse masked modeling for vision decoding," in *Proceedings of the IEEE/CVF Conference on Computer Vision and Pattern Recognition (CVPR)*, 2023, pp. 22 710–22 720.

[22] J. Sun, M. Li, Z. Chen, Y. Zhang, S. Wang, and M.-F. Moens, "Contrast, attend and diffuse to decode high-resolution images from brain activities," *Advances in Neural Information Processing Systems (NeurIPS) (NeurIPS)*, vol. 36, 2024.

[23] B. Zeng, S. Li, X. Liu, S. Gao, X. Jiang, X. Tang, Y. Hu, J. Liu, and B. Zhang, "Controllable mind visual diffusion model," in *Proceedings of the AAAI Conference on Artificial Intelligence*, vol. 38, no. 7, 2024, pp. 6935–6943.

[24] P. Scotti, A. Banerjee, J. Goode, S. Shabalin, A. Nguyen, A. Dempster, N. Verlinde, E. Yundler, D. Weisberg, K. Norman *et al.*, "Reconstructing the mind's eye: fmri-to-image with contrastive learning and diffusion priors," *Advances in Neural Information Processing Systems (NeurIPS)*, vol. 36, 2024.

[25] W. Xia, R. de Charette, C. Oztireli, and J.-H. Xue, "Dream: Visual decoding from reversing human visual system," in *Proceedings of the IEEE/CVF Winter Conference on Applications of Computer Vision (WACV)*, 2024, pp. 8226–8235.

[26] F. Ozcelik and R. VanRullen, "Natural scene reconstruction from fmri signals using generative latent diffusion," *Scientific Reports*, vol. 13, no. 1, p. 15666, 2023.

[27] P. S. Scotti, M. Tripathy, C. K. T. Villanueva, R. Kneeland, T. Chen, A. Narang, C. Santhirasegaran, J. Xu, T. Naselaris, K. A. Norman *et al.*, "Mindeye2: Shared-subject models enable fmri-to-image with 1 hour of data," *arXiv preprint arXiv:2403.11207*, 2024.

[28] Y. Benchetrit, H. Banville, and J.-R. King, "Brain decoding: toward real-time reconstruction of visual perception," in *The Twelfth International Conference on Learning Representations (ICLR)*, 2024. [Online]. Available: https://openreview.net/forum?id=3y1K6buO8c

[29] Y. Bai, X. Wang, Y.-p. Cao, Y. Ge, C. Yuan, and Y. Shan, "Dreamdiffusion: Generating high-quality images from brain EEG signals," *arXiv preprint arXiv:2306.16934*, 2023.

[30] Y.-T. Lan, K. Ren, Y. Wang, W.-L. Zheng, D. Li, B.-L. Lu, and L. Qiu, "Seeing through the brain: image reconstruction of visual perception from human brain signals," *arXiv preprint arXiv:2308.02510*, 2023.

[31] Z. Chen, J. Qing, and J. H. Zhou, "Cinematic mindscapes: High-quality video reconstruction from brain activity," *Advances in Neural Information Processing Systems (NeurIPS)*, vol. 36, 2024.

[32] J. Sun, M. Li, Z. Chen, and M.-F. Moens, "Neurocine: Decoding vivid video sequences from human brain activties," *arXiv preprint arXiv:2402.01590*, 2024.

[33] J. Z. Wu, Y. Ge, X. Wang, S. W. Lei, Y. Gu, Y. Shi, W. Hsu, Y. Shan, X. Qie, and M. Z. Shou, "Tune-a-video: One-shot tuning of image diffusion models for text-to-video generation," in *Proceedings of the IEEE/CVF International Conference on Computer Vision (ICCV)*, 2023, pp. 7623–7633.

[34] Y. Miyawaki, H. Uchida, O. Yamashita, M.-a. Sato, Y. Morito, H. C. Tanabe, N. Sadato, and Y. Kamitani, "Visual image reconstruction from human brain activity using a combination of multiscale local image decoders," *Neuron*, vol. 60, no. 5, pp. 915–929, 2008.

[35] T. Naselaris, R. J. Prenger, K. N. Kay, M. Oliver, and J. L. Gallant, "Bayesian reconstruction of natural images from human brain activity," *Neuron*, vol. 63, no. 6, pp. 902–915, 2009.

[36] G. J. Brouwer and D. J. Heeger, "Decoding and reconstructing color from responses in human visual cortex," *Journal of Neuroscience*, vol. 29, no. 44, pp. 13 992–14 003, 2009.

[37] S. Schoenmakers, M. Barth, T. Heskes, and M. Van Gerven, "Linear reconstruction of perceived images from human brain activity," *NeuroImage*, vol. 83, pp. 951–961, 2013.

[38] R. Beliy, G. Gaziv, A. Hoogi, F. Strappini, T. Golan, and M. Irani, "From voxels to pixels and back: Self-supervision in natural-image reconstruction from fmri," *Advances in Neural Information Processing Systems (NeurIPS)*, vol. 32, 2019.

[39] G. Gaziv, R. Beliy, N. Granot, A. Hoogi, F. Strappini, T. Golan, and M. Irani, "Self-supervised natural image reconstruction and large-scale semantic classification from brain activity," *NeuroImage*, vol. 254, p. 119121, 2022.

[40] S. Lin, T. Sprague, and A. K. Singh, "Mind reader: Reconstructing complex images from brain activities," *Advances in Neural Information Processing Systems (NeurIPS)*, vol. 35, pp. 29 624–29 636, 2022.

[41] F. Ozcelik, B. Choksi, M. Mozafari, L. Reddy, and R. VanRullen, "Reconstruction of perceived images from fmri patterns and semantic brain exploration using instance-conditioned gans," in *2022 International Joint Conference on Neural Networks (IJCNN)*. IEEE, 2022, pp. 1–8.

[42] L. Meng and C. Yang, "Semantics-guided hierarchical feature encoding generative adversarial network for visual image reconstruction from brain activity," *IEEE Transactions on Neural Systems and Rehabilitation Engineering*, vol. 32, pp. 1267–1283, 2024.

[43] S. Palazzo, C. Spampinato, I. Kavasidis, D. Giordano, and M. Shah, "Generative adversarial networks conditioned by brain signals," in *Proceedings of the IEEE International Conference on Computer Vision (ICCV)*, 2017, pp. 3410–3418.

[44] I. Kavasidis, S. Palazzo, C. Spampinato, D. Giordano, and M. Shah, "Brain2image: Converting brain signals into images," in *Proceedings of the 25th ACM International Conference on Multimedia*, 2017, pp. 1809–1817.

[45] R. Rombach, A. Blattmann, D. Lorenz, P. Esser, and B. Ommer, "High-resolution image synthesis with latent diffusion models," in *Proceedings of the IEEE/CVF Conference on Computer Vision and Pattern Recognition (CVPR)*, 2022, pp. 10 684–10 695.

[46] G. Kim, T. Kwon, and J. C. Ye, "Diffusionclip: Text-guided diffusion models for robust image manipulation," in *Proceedings of the IEEE/CVF Conference on Computer Vision and Pattern Recognition (CVPR)*, 2022, pp. 2426–2435.

[47] J. Ho, A. Jain, and P. Abbeel, "Denoising diffusion probabilistic models," *Advances in Neural Information Processing Systems (NeurIPS)*, vol. 33, pp. 6840–6851, 2020.

[48] A. Radford, J. W. Kim, C. Hallacy, A. Ramesh, G. Goh, S. Agarwal, G. Sastry, A. Askell, P. Mishkin, J. Clark *et al.*, "Learning transferable visual models from natural language supervision," in *International Conference on Machine Learning (ICML)*. PMLR, 2021, pp. 8748–8763.

[49] C. Spampinato, S. Palazzo, I. Kavasidis, D. Giordano, N. Souly, and M. Shah, "Deep learning human mind for automated visual classification," in *Proceedings of the IEEE/CVF Conference on Computer Vision and Pattern Recognition (CVPR)*, 2017, pp. 6809–6817.

[50] R. Li, J. S. Johansen, H. Ahmed, T. V. Ilyevsky, R. B. Wilbur, H. M. Bharadwaj, and J. M. Siskind, "The perils and pitfalls of block design for EEG classification experiments," *IEEE Transactions on Pattern Analysis and Machine Intelligence*, vol. 43, no. 1, pp. 316–333, 2020.

[51] H. Ahmed, R. B. Wilbur, H. M. Bharadwaj, and J. M. Siskind, "Object classification from randomized EEG trials," in *Proceedings of the IEEE/CVF Conference on Computer Vision and Pattern Recognition (CVPR)*, 2021, pp. 3845–3854.

[52] H. Wen, J. Shi, Y. Zhang, K.-H. Lu, J. Cao, and Z. Liu, "Neural encoding and decoding with deep learning for dynamic natural vision," *Cerebral Cortex*, vol. 28, no. 12, pp. 4136–4160, 2018.

[53] G. Kupershmidt, R. Beliy, G. Gaziv, and M. Irani, "A penny for your (visual) thoughts: Self-supervised reconstruction of natural movies from brain activity," *arXiv preprint arXiv:2206.03544*, 2022.

[54] C. Wang, H. Yan, W. Huang, J. Li, Y. Wang, Y.-S. Fan, W. Sheng, T. Liu, R. Li, and H. Chen, "Reconstructing rapid natural vision with fmri-conditional video generative adversarial network," *Cerebral Cortex*, vol. 32, no. 20, pp. 4502–4511, 2022.

[55] J. Li, D. Li, C. Xiong, and S. Hoi, "Blip: Bootstrapping language-image pre-training for unified vision-language understanding and generation," in *International Conference on Machine Learning (ICML)*. PMLR, 2022, pp. 12 888–12 900.

[56] H. Intraub, "Rapid conceptual identification of sequentially presented pictures." *Journal of Experimental Psychology: Human Perception and Performance*, vol. 7, no. 3, p. 604, 1981.

[57] K. N. Kay, T. Naselaris, R. J. Prenger, and J. L. Gallant, "Identifying natural images from human brain activity," *Nature*, vol. 452, no. 7185, pp. 352–355, 2008.

[58] V. J. Lawhern, A. J. Solon, N. R. Waytowich, S. M. Gordon, C. P. Hung, and B. J. Lance, "EEGnet: a compact convolutional neural network for EEG-based brain–computer interfaces," *Journal of neural engineering*, vol. 15, no. 5, p. 056013, 2018.

[59] Y. Song, Q. Zheng, B. Liu, and X. Gao, "EEG conformer: Convolutional transformer for EEG decoding and visualization," *IEEE Transactions on Neural Systems and Rehabilitation Engineering*, vol. 31, pp. 710–719, 2022.

[60] A. Vaswani, N. Shazeer, N. Parmar, J. Uszkoreit, L. Jones, A. N. Gomez, Ł. Kaiser, and I. Polosukhin, "Attention is all you need," *Advances in neural information processing systems*, vol. 30, 2017.

[61] A. Hyvärinen, "New approximations of differential entropy for independent component analysis and projection pursuit," *Advances in Neural Information Processing Systems (NeurIPS)*, vol. 10, 1997.

[62] R. T. Schirrmeister, J. T. Springenberg, L. D. J. Fiederer, M. Glasstetter, K. Eggensperger, M. Tangermann, F. Hutter, W. Burgard, and T. Ball, "Deep learning with convolutional neural networks for EEG decoding and visualization," *Human brain mapping*, vol. 38, no. 11, pp. 5391–5420, 2017.

[63] C. Cortes and V. Vapnik, "Support-vector networks," *Machine learning*, vol. 20, pp. 273–297, 1995.

[64] P. Bao, L. She, M. McGill, and D. Y. Tsao, "A map of object space in primate inferotemporal cortex," *Nature*, vol. 583, no. 7814, pp. 103–108, 2020.

[65] J. H. Maunsell and D. C. Van Essen, "Functional properties of neurons in middle temporal visual area of the macaque monkey. ii. binocular interactions and sensitivity to binocular disparity," *Journal of neurophysiology*, vol. 49, no. 5, pp. 1148–1167, 1983.

[66] Z. Wang, A. C. Bovik, H. R. Sheikh, and E. P. Simoncelli, "Image quality assessment: from error visibility to structural similarity," *IEEE transactions on image processing*, vol. 13, no. 4, pp. 600–612, 2004.

[67] J. Deng, W. Dong, R. Socher, L.-J. Li, K. Li, and L. Fei-Fei, "Imagenet: A large-scale hierarchical image database," in *2009 IEEE conference on computer vision and pattern recognition*.   Ieee, 2009, pp. 248–255.

[68] Z. Tong, Y. Song, J. Wang, and L. Wang, "Videomae: Masked autoencoders are data-efficient learners for self-supervised video pre-training," *Advances in Neural Information Processing Systems (NeurIPS)*, vol. 35, pp. 10 078–10 093, 2022.

[69] W. Kay, J. Carreira, K. Simonyan, B. Zhang, C. Hillier, S. Vijayanarasimhan, F. Viola, T. Green, T. Back, P. Natsev *et al.*, "The kinetics human action video dataset," *arXiv preprint arXiv:1705.06950*, 2017.

[70] J. Song, C. Meng, and S. Ermon, "Denoising diffusion implicit models," *arXiv preprint arXiv:2010.02502*, 2020.

[71] E. Orchard-Mills, D. Alais, and E. Van der Burg, "Cross-modal associations between vision, touch, and audition influence visual search through top-down attention, not bottom-up capture," *Attention, Perception, & Psychophysics*, vol. 75, pp. 1892–1905, 2013.

[72] J. Gu, B. Liu, X. Li, P. Wang, and B. Wang, "Cross-modal representations in early visual and auditory cortices revealed by multi-voxel pattern analysis," *Brain Imaging and Behavior*, vol. 14, no. 5, pp. 1908–1920, 2020.

[73] V. Marian, S. Hayakawa, and S. R. Schroeder, "Cross-modal interaction between auditory and visual input impacts memory retrieval," *Frontiers in Neuroscience*, vol. 15, p. 661477, 2021.

# A Evaluation Metrics for Video Reconstruction Benchmark

In this section, we detail the metrics we use for the video reconstruction benchmark. The metrics to evaluate the quality of generated videos can be roughly classified as frame-based metrics and video-based metrics.

**Frame-based Metrics** Two levels of metrics are considered to judge the quality of generated frames: the pixel-level and the semantics-level metrics. For the pixel level, we calculate the average structural similarity index measure (SSIM) [66] of each frame between the ground-truth video and the reconstructed video. For the semantic level, a CLIP-based classifier [48] trained on ImageNet [67] is adopted to compute the $N$-way top-$K$ accuracy of predicted frames. If the ground-truth class is within the top-$K$ probability of the predicted frames classification results from $N$ arbitrary classes (including ground-truth class), the semantic-level reconstruction is regarded successful.

**Video-based Metric** As the ImageNet classifier is unable to well understand videos, a VideoMAE-based [68] video classifier trained on Kinetics-400 dataset [69], which can understand 400 dynamic concepts (e.g., changes, human motions), is applied to compute the video semantic-level accuracy.

# B Experiment Setup and Implementation Details

## B.1 EEG Signals Preprocessing

The raw data was recorded with the 62-channel EEG cap with a sample rate of 1000 Hz and stored in the continuous EEG data file format (.cnt), a single file for the experiment of each subject. We applied the 0.1-100 Hz band-pass filter to filter out the DC interference and very high-frequency interference and down-sampled the EEG data to 200Hz to accelerate computations. A one-second sliding window with 500 ms overlapping is used for EEG segmentation and frequency feature extraction. Specifically, power spectral density (PSD) and differential entropy (DE) [61] of five frequency bands ($\delta$: 1-4 Hz, $\theta$: 4-8 Hz, $\alpha$: 8-14 Hz, $\beta$: 14-31 Hz, and $\gamma$: 31-99 Hz) are extracted.

## B.2 Classification Experiment Details

We perform a 7-fold cross-validation and report the average accuracy. Specifically, we select each single video block as the testing set one by one, the block before testing set as the validation set, and the remaining 5 blocks compose the training set. Several EEG classifiers, including ShallowNet, DeepNet [62], EEGNet [58], Conformer [59], and TSConv [19], are adopted as baselines for processing raw EEG. For frequency features, we compare the performance of support vector machine (SVM) [63] and multilayer perceptron (MLP). In our experiment, Our GLMNet uses a concise global encoder, which has the same architecture of ShallowNet on raw signals, and MLP on frequency featrues. All models are implemented with PyTorch and evaluated on an Nvidia A100 GPU. Adam optimizer is used with the learning rate $\eta = 0.001$. Batch size is set to 256 for all methods, and the number of training epochs is 100.

## B.3 Video Reconstruction Experiment Details

We reconstruct a two-second video clip from the corresponding two-second EEG segment. For efficient training and testing, we down-sampled the 24 FPS 1080p original videos to a small video of resolution of $512 \times 288$ (16:9) with 3 FPS, resulting in 6 frames for each video. A 500 ms sliding window with 250 ms overlapping is used for EEG segmentation, forming a total of 7 segments in an EEG embeddings $e_{eeg} \in \mathbb{R}^{7 \times d}$, where $d$ is the embedding dimension and we set $d = 512$.

We use the first 6 blocks from all the sessions as the training set and the last blocks as the testing set in our experiment. The Transformer used as the Seq2Seq model has 2 encoder layers and 4 decoder layers. Semantic predictor has 4 layers with ReLU activation. Dynamic predictor is GLMNet. An Adam optimizer with learning rate of 0.0005 and cosine scheduler was adopted for training the above models with 200 epochs. The inflated video generation model is fine-tuned using the same setting in [33] on the training set with learning rate of 0.00003 and cosine scheduler for 200 epochs, which takes about 5 hours. The inference is performed with 100 DDIM [70] steps.

Limited to the low signal-to-noise ratio (SNR) and spatial resolution, decoding videos from EEG is somehow difficult. Thus, besides applying the full dataset of 40 concepts, we select {*Cat*, *Shark*,

*Flower*, *Dancing*, *Face*, *Buildings*, *Road*, *Pizza*, *Guitar*, *Airplane*} to form a 10-class subset, and the first {1-20} categories and the first {1-30} categories to form other two subsets whose sizes are 20 and 30.

## C  More Results on EEG-VP benchmark

### C.1  40-class Classification Task

We plot the confusion matrices of GLMNet on the 40-class task. It can be seen that though the accuracy is not high, there is a faint diagonal lines in both confusion matrices. Moreover, a small square in the right bottom corner is being observed, of which categories are {*Drum*, *Guitar*, and *Piano*} (32 - 34 class). This discovery may caused by the reason that the musical instruments stimulate the auditory cortex in our brains with these visual cues, a well-studied phenomenon named cross-modal perception[71–73], which then has similar reflections in EEG signals.

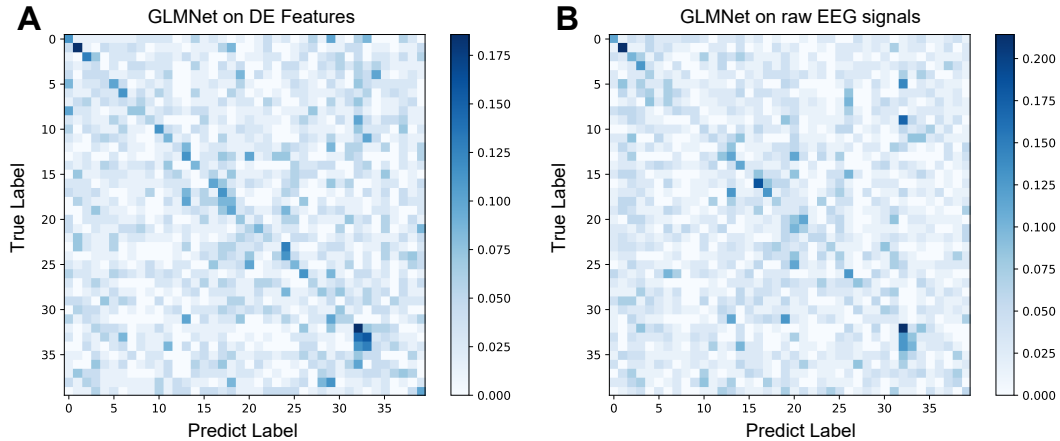

Figure 6: **Confusion Matrices of GLMNet** (A). The performance using DE features. (B). The performance using raw EEG signals.

### C.2  Performance of Each Subject

EEG signal is notorious for its large variations among different subjects. In order to verify whether the dynamic visual perception decoding generalize across different people, we plot the results on the EEG-VP benchmark of all the subjects separately in Figure 7. Generally speaking, model trained on all the individual subject can achieve reasonable results across different tasks, demonstrating that the phenomenon of visual perception information being contained in the EEG is common across different people. Meanwhile, we can see the differences in visual specificity contained in the EEG signals among different subjects. For example, EEG from Subject 3 predicts well on the class of the objects, EEG from Subject 4 and 13 are more sensitive to color information, and EEG from Subject 9 is better at capturing motor information.

## D  More Reconstructed Samples

We display more reconstruction results for demonstrating the effectiveness of our EEG2Video framework. It can be observed that various videos across animals, plants, people, and activities. Please kindly refer to the supplementary files to find GIFs of these reconstructed videos.

Some failure samples are displayed in Figure 13. These failures are typically caused by the inability of the model to infer either the semantic information or the low-level visual information correctly, resulting the irrelevantly generated videos. However, we can still see from these failed examples that the model reconstructs some features of the real video, from shapes, movements, to the scene dynamics of the video. For example, the man practicing boxing is reconstructed as a panda practicing

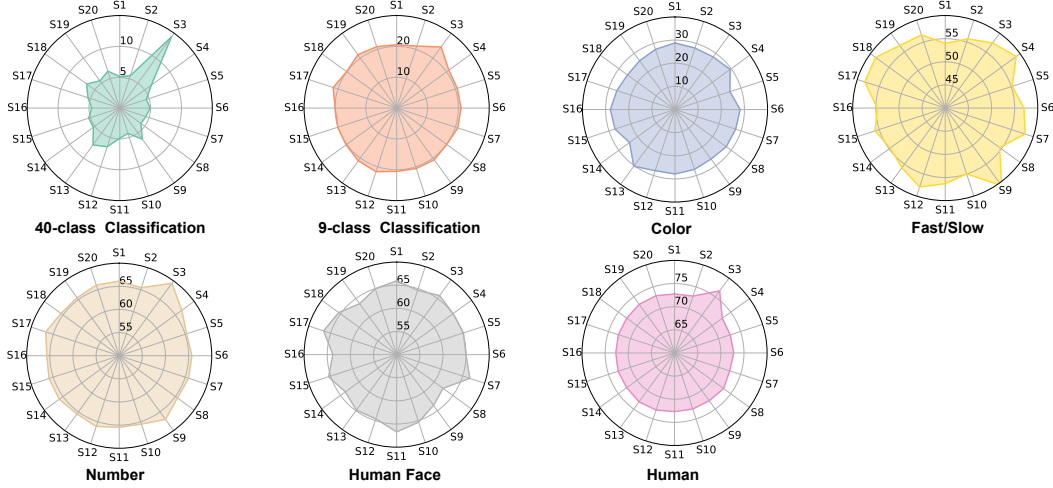

Figure 7: Performance of each subject on different tasks on EEG-VP benchmark.

boxing, the ship sailing on the sea is reconstructed as a shark, and the fast-moving car is reconstructed as a fast-moving person.

# E  Algorithm of EEG2Video

In this section, we write the algorithm of EEG2Video framework in Algorithm 1.

---

**Algorithm 1** Training Stage of EEG2Video Framework

---

**Input:** (1) training set $\mathcal{D}_{train} = \{x_i, v_i, d_i\}$, where $x_i$ is EEG segments, $v_i$ is video clips, $d_i$ is the *fast/slow* label (2) stable diffusion model $T2I$, whose VAE encoder is $\mathcal{E}_{vae}$ (3) image caption model $B$

**Output:** (1) video diffusion model $T2V$, (2) Seq2Seq model $Seq2Seq$, (3) semantic predictor $\mathcal{P}_s$, (4) dynamic predictor $\mathcal{P}_d$

1: Initialize text prompts of training dataset $T = \{t_i\}$
2: **for** each $(v_i) \in \mathcal{D}_{train}$ **do**
3:     $v_i = \{f_1, f_2, \ldots, f_n\}$
4:     $t_i \leftarrow B(f_1)$
5: **end for**
6: Initialize latent vectors of all frames $L = \{z_i\}$
7: **for** each $(v_i) \in \mathcal{D}_{train}$ **do**
8:     $v_i = \{f_1, f_2, \ldots, f_n\}$
9:     $z_i = \{l_1, l_2, \ldots, l_n\}$
10:     **for** each $(f_j) \in v_i$ **do**
11:         $l_j \leftarrow VAE(f_j)$
12:     **end for**
13: **end for**
14: Fine-tune the $T2I$ with $\{v_i, t_i\}$ to obtain video diffusion model $T2V$
15: Train the Seq2Seq model $Seq2Seq$ with all $\{x_i, z_i\}$ using MSE loss
16: Train the semantic predictor $\mathcal{P}_s$ with all $\{x_i, t_i\}$ using MSE loss
17: Train the dynamic predictor $\mathcal{P}_d$ with all $\{x_i, d_i\}$ using Cross Entropy loss
18: **return** $T2V, Seq2Seq, \mathcal{P}_s, \mathcal{P}_d$

---

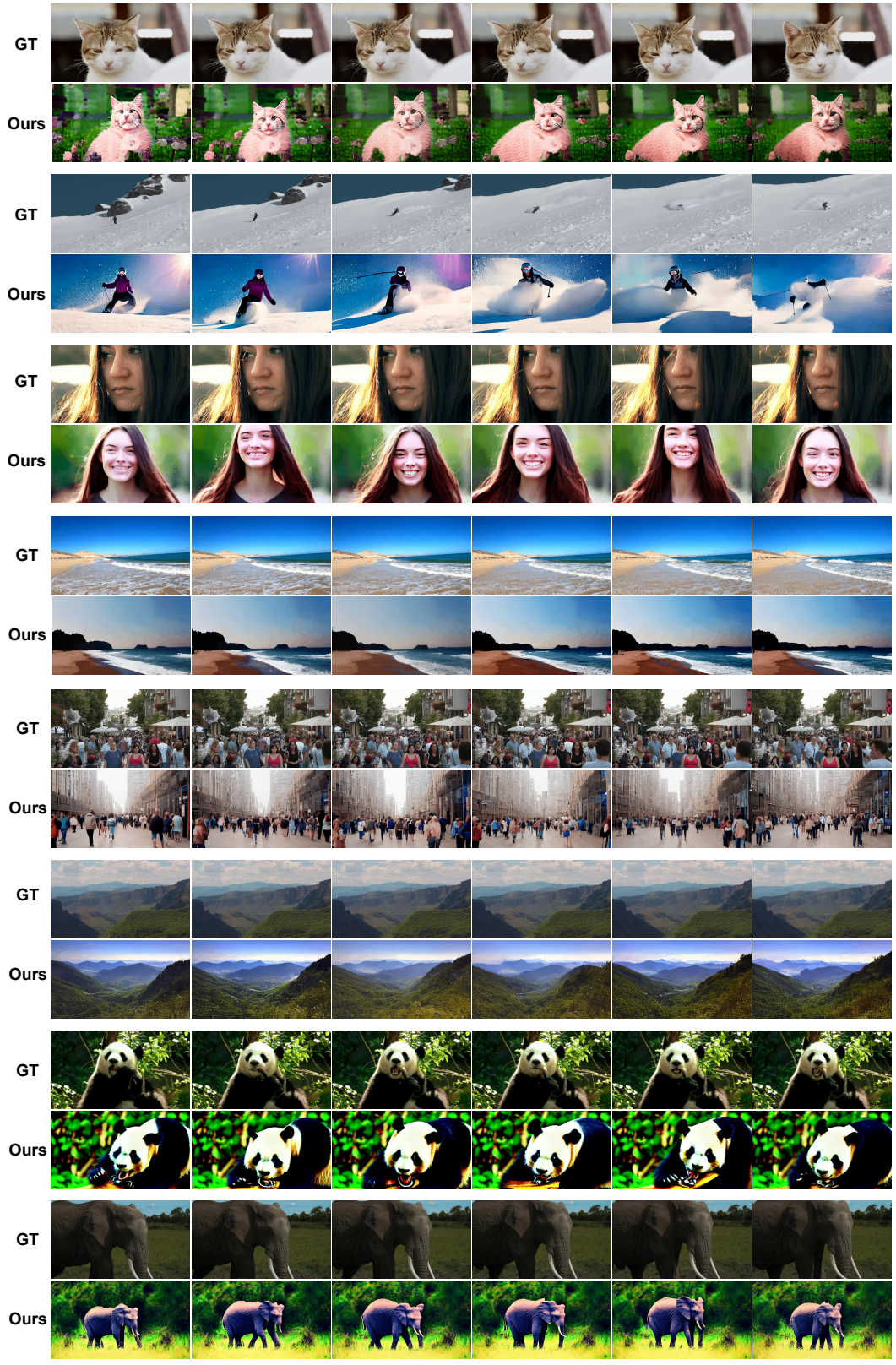

Figure 8: Various videos reconstruction samples.

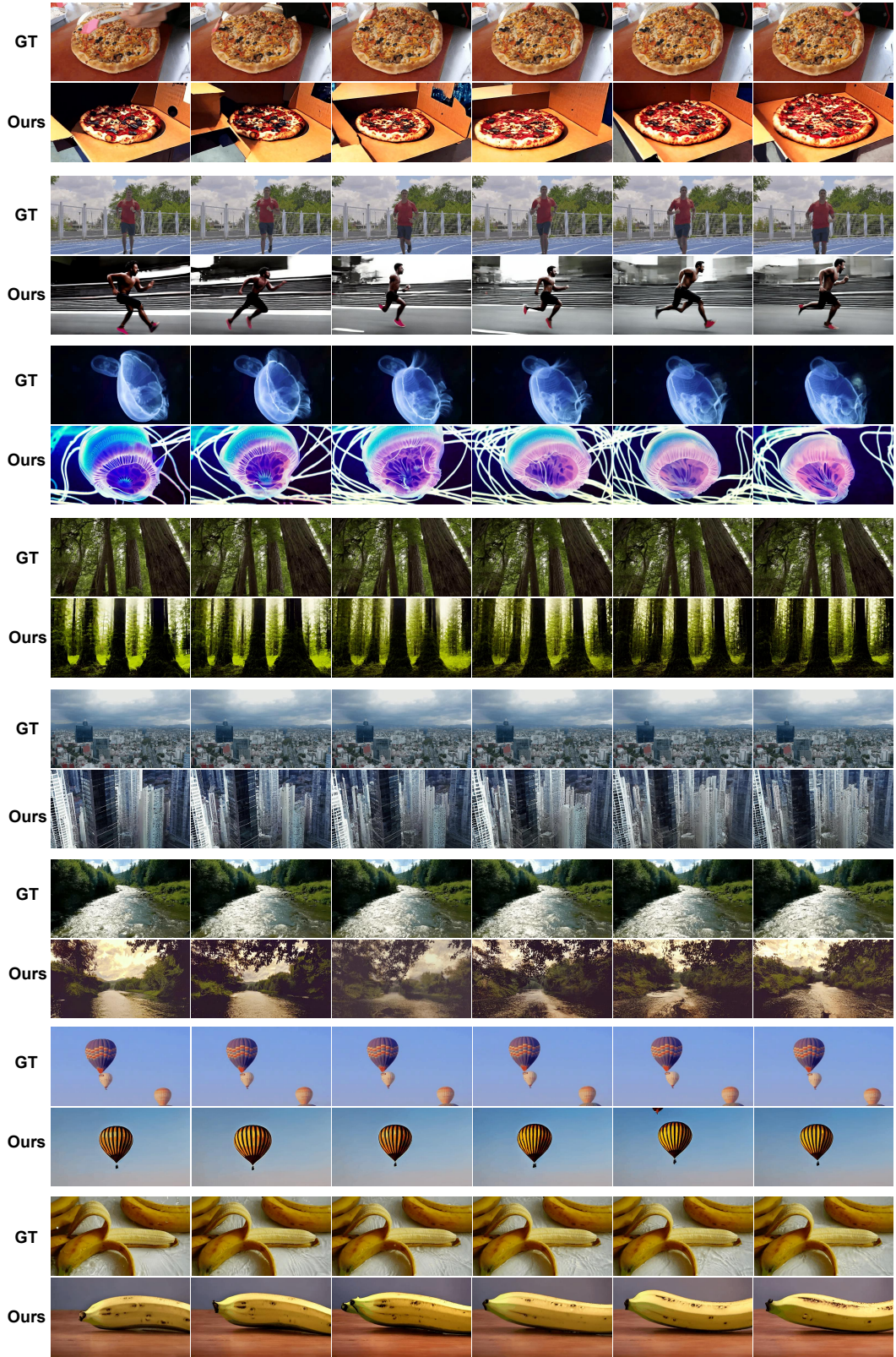

Figure 9: Various videos reconstruction samples.

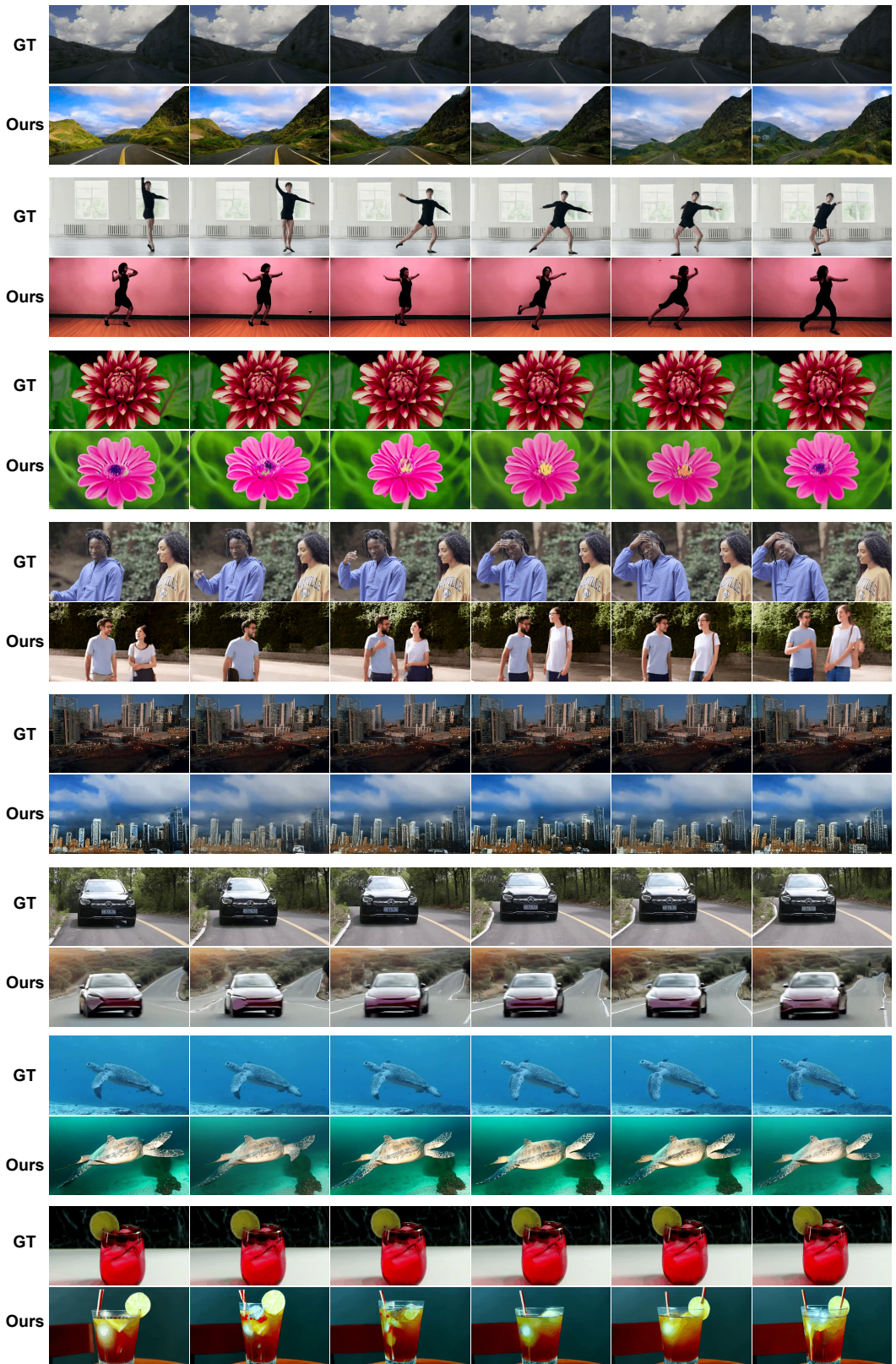

Figure 10: Various videos reconstruction samples.

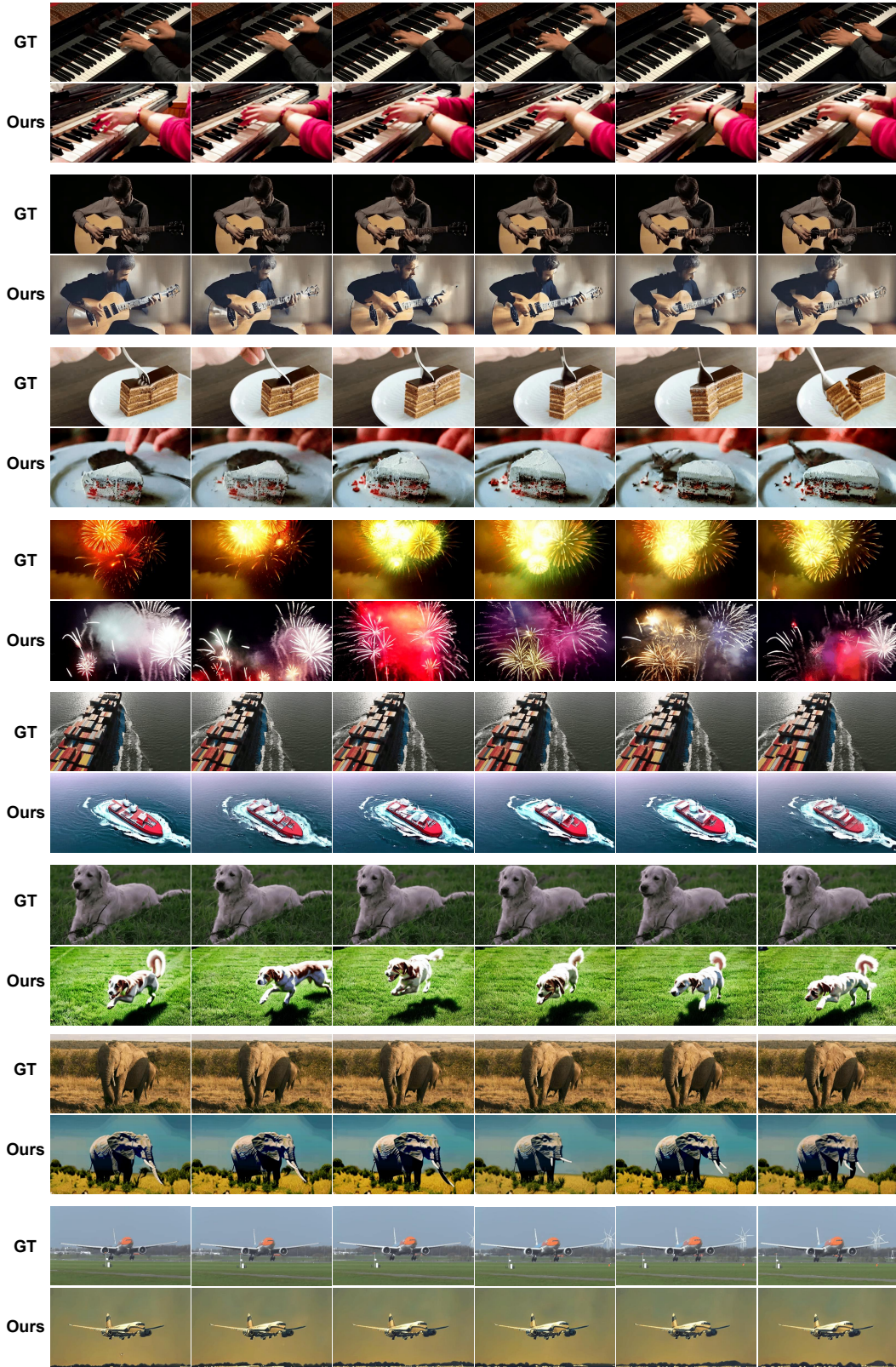

Figure 11: Various videos reconstruction samples.

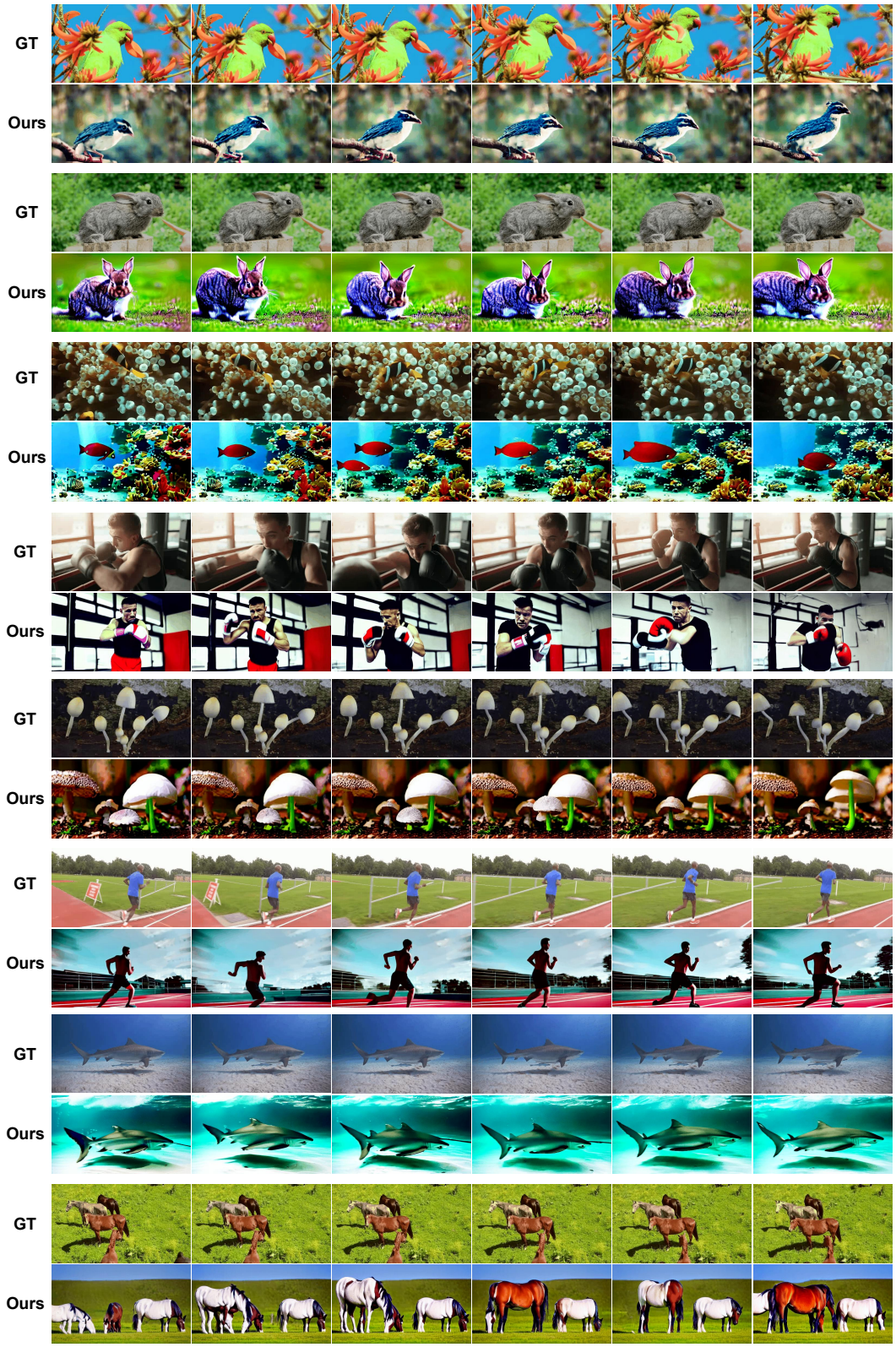

Figure 12: Various videos reconstruction samples.

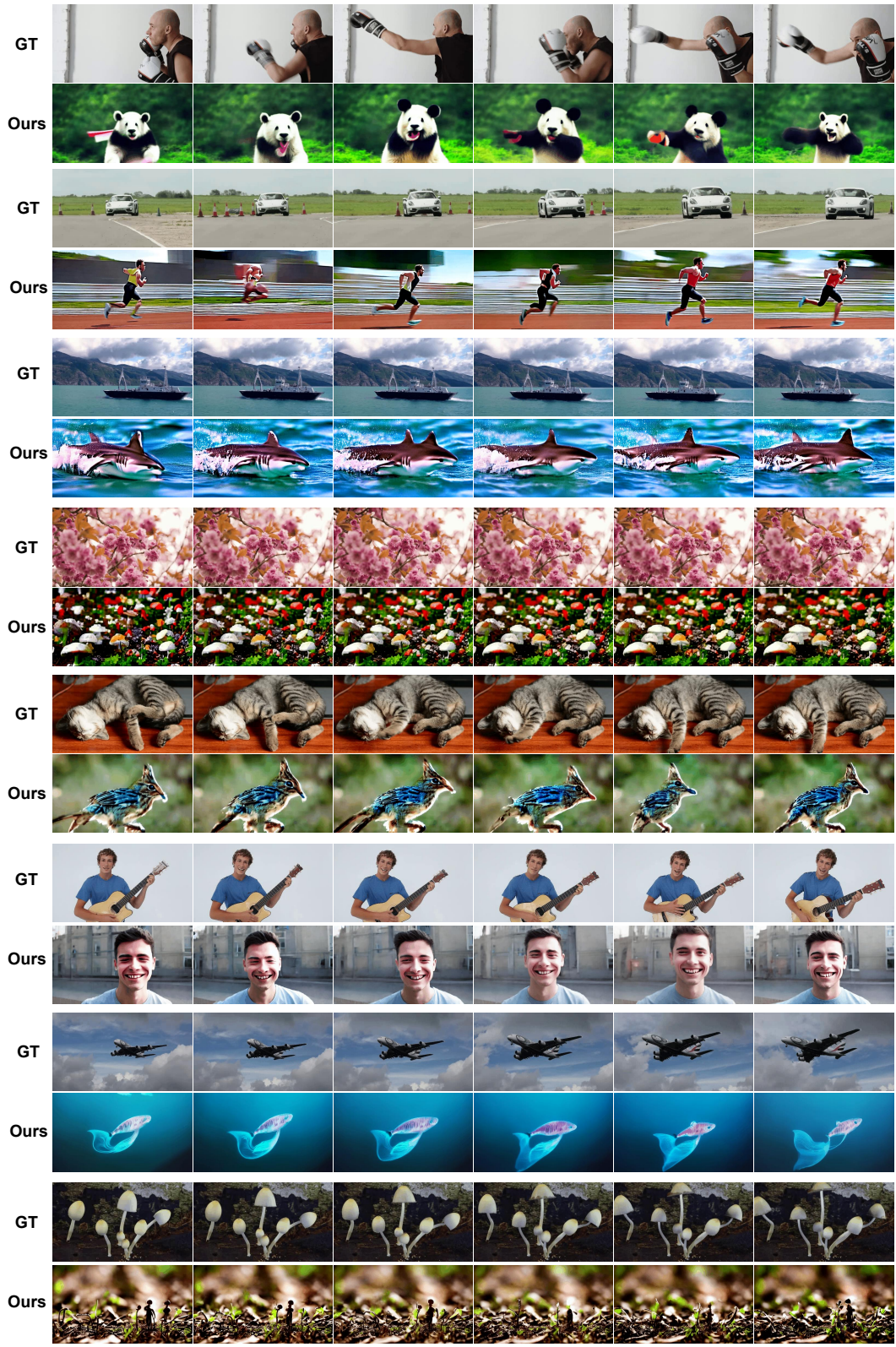

Figure 13: Some failure samples reconstructed in ours.

